# Adaptive Bayesian Inference

**Umut A. Acar**[*]
Toyota Tech. Inst.
Chicago, IL
*umut@tti-c.org*

**Alexander T. Ihler**
U.C. Irvine
Irvine, CA
*ihler@ics.uci.edu*

**Ramgopal R. Mettu**[†]
Univ. of Massachusetts
Amherst, MA
*mettu@ecs.umass.edu*

**Özgür Sümer**
Uni. of Chicago
Chicago, IL
*osumer@cs.uchicago.edu*

## Abstract

Motivated by stochastic systems in which observed evidence and conditional dependencies between states of the network change over time, and certain quantities of interest (marginal distributions, likelihood estimates etc.) must be updated, we study the problem of *adaptive inference* in tree-structured Bayesian networks. We describe an algorithm for adaptive inference that handles a broad range of changes to the network and is able to maintain marginal distributions, MAP estimates, and data likelihoods in all expected logarithmic time. We give an implementation of our algorithm and provide experiments that show that the algorithm can yield up to two orders of magnitude speedups on answering queries and responding to dynamic changes over the sum-product algorithm.

## 1 Introduction

Graphical models [14, 8] are a powerful tool for probabilistic reasoning over sets of random variables. Problems of inference, including marginalization and MAP estimation, form the basis of statistical approaches to machine learning. In many applications, we need to perform inference under dynamically changing conditions, such as the acquisition of new evidence or an alteration of the conditional relationships which make up the model. Such changes arise naturally in the experimental setting, where the model quantities are empirically estimated and may change as more data are collected, or in which the goal is to assess the effects of a large number of possible interventions. Motivated by such applications, Delcher *et al.* [6] identify *dynamic Bayesian inference* as the problem of performing Bayesian inference on a dynamically changing graphical model. Dynamic changes to the graphical model may include changes to the observed evidence, to the structure of the graph itself (such as edge or node insertions/deletions), and changes to the conditional relationships among variables.

To see why adapting to dynamic changes is difficult, consider the simple problem of Bayesian inference in a Markov chain with $n$ variables. Suppose that all marginal distributions have been computed in $O(n)$ time using the sum-product algorithm, and that some conditional distribution at a node $u$ is subsequently updated. One way to update the marginals would be to recompute the messages computed by sum-product from $u$ to other nodes in the network. This can take $\Omega(n)$ time because regardless of where $u$ is in the network, there always is another node $v$ at distance $\Omega(n)$ from $u$. A similar argument holds for general tree-structured networks. Thus, simply updating sum-product messages can be costly in applications where marginals must be adaptively updated after changes to the model (see Sec. 5 for further discussion).

In this paper, we present a technique for efficient adaptive inference on graphical models. For a tree-structured graphical model with $n$ nodes, our approach supports the computation of any marginal, updates to conditional probability distributions (including observed evidence) and edge insertions

---

[*]U. A. Acar is supported by a gift from Intel.

[†]R. R. Mettu is supported by a National Science Foundation CAREER Award (IIS-0643768).

and deletions in expected $O(\log n)$ time. As an example of where adaptive inference can be effective, consider a computational biology application that requires computing the state of the active site in a protein as the user modifies the protein (e.g., mutagenesis). In this application, we can represent the protein with a graphical model and use marginal computations to determine the state of the active site. We reflect the modifications to the protein by updating the graphical model representation and performing marginal queries to obtain the state of the active site. We show in Sec. 5 that our approach can achieve a speedup of one to two orders of magnitude over the sum-product algorithm in such applications.

Our approach achieves logarithmic update and query times by mapping an arbitrary tree-structued graphical model into a balanced representation that we call a *cluster tree* (Sec. 3–4). We perform an $O(n)$-time preprocessing step to compute the *cluster tree* using a technique known as *tree contraction* [13]. We ensure that for an input network with $n$ nodes, the cluster tree has an expected depth of $O(\log n)$ and expected size $O(n)$. We show that the nodes in the cluster tree can be tagged with partial computations (corresponding to marginalizations of subtrees of the input network) in way that allows marginal computations and changes to the network to be performed in $O(\log n)$ expected time. We give simulation results (Sec. 5) that show that our algorithm can achieve a speedup of one to two orders of magnitude over the sum-product algorithm. Although we focus primarily on the problem of answering marginal queries, it is straightforward to generalize our algorithms to other, similar inference goals, such as MAP estimation and evaluating the likelihood of evidence.

We note that although tree-structured graphs provide a relatively restrictive class of models, junction trees [14] can be used to extend some of our results to more general graphs. In particular, we can still support changes to the parameters of the distribution (evidence and conditional relationships), although changes to the underlying graph structure become more difficult. Additionally, a number of more sophisticated graphical models require efficient inference over trees at their core, including learning mixtures of trees [12] and tree-reparameterized max-product [15]. Both these methods involve repeatedly performing a message passing algorithm over a set of trees with changing parameters or evidence, making efficient updates and recomputations a significant issue.

**Related Work.** It is important to contrast our notion of adapting to dynamic updates to the graphical model (due to changes in the evidence, or alterations of the structure and distribution) with the potentially more general definition of dynamic Bayes' nets (DBNs) [14]. Specifically, a DBN typically refers to a Bayes' net in which the variables have an explicit notion of time, and past observations have some influence on estimates about the present and future. Marginalizing over unobserved variables at time $t-1$ typically produces increased complexity in the the model of variables at time $t$. However, in both [6] and this work, the emphasis is on performing inference with *current* information only, and efficiency is obtained by leveraging the similarity between the previous and newly updated models.

Our work builds on previous work by Delcher, Grove, Kasif and Pearl [6]; they give an algorithm to update Bayesian networks dynamically as the observed variables in the network change and compute belief queries of hidden variables in logarithmic time. The key difference between their work and ours is that their algorithm only supports updates to observed evidence, and does not support dynamic changes to the graph structure (i.e., insertion/deletion of edges) or to conditional probabilities. In many applications it is important to consider the effect of changes to conditional relationships between variables; for example, to study protein structure (see Sec. 5 for further discussion). In fact, Delcher *et al.* cite structural updates to the given network as an open problem. Another difference includes the use of tree contraction: they use tree contractions to answer queries and perform updates. We use tree contractions to construct a cluster tree, which we then use to perform queries and all other updates (except for insertions/deletions). We provide an implementation and show that this approach yields significant speedups.

Our approach to clustering factor graphs using tree contractions is based on previous results that show that tree contractions can be updated in expected logarithmic time under certain dynamic changes by using a general-purpose change-propagation algorithm [2]. The approach has also been applied to a number of basic problems on trees [3] but has not been considered in the context of statistical inference. The change-propagation approach used in this work has also been extended to provide a general-purpose technique for updating computations under changes to their data and applied to a number of applications (e.g. [1]).

## 2 Background

Graphical models provide a convenient formalism for describing the structure of a function $g$ defined over a set of variables $x_1, \ldots, x_n$ (most commonly a joint probability distribution over the $x_i$). Graphical models use this structure to organize computations and create efficient algorithms for many inference tasks over $g$, such as finding a maximum a-posteriori (MAP) configuration, marginalization, or computing data likelihood. For the purposes of this paper, we assume that each variable $x_i$ takes on values from some finite set, denoted $A_i$. We write the operation of marginalizing over $x_i$ as $\sum_{x_i}$, and let $X_j$ represent an index-ordered subset of variables and $f(X_j)$ a function defined over those variables, so that for example if $X_j = \{x_2, x_3, x_5\}$, then the function $f(X_j) = f(x_2, x_3, x_5)$. We use $X$ to indicate the index-ordered set of all $\{x_1, \ldots, x_n\}$.

**Factor Graphs.** A factor graph [10] is one type of graphical model, similar to a Bayes' net [14] or Markov random field [5] used to represent the factorization structure of a function $g(x_1, \ldots, x_n)$. In particular, suppose that $g$ decomposes into a product of simpler functions, $g(X) = \prod_j f_j(X_j)$, for some collection of real-valued functions $f_j$, called *factors*, whose arguments are (index-ordered) sets $X_j \subseteq X$. A factor graph consists of a graph-theoretic abstraction of $g$'s factorization, with vertices of the graph representing variables $x_i$ and factors $f_j$. Because of the close correspondence between these quantities, we abuse notation slightly and use $x_i$ to indicate both the variable and its associated vertex, and $f_j$ to indicate both the factor and its vertex.

**Definition 2.1.** *A* **factor graph** *is a bipartite graph* $G = (X + F, E)$ *where* $X = \{x_1, x_2, \ldots, x_n\}$ *is a set of variables,* $F = \{f_1, f_2, \ldots, f_m\}$ *is a set of factors and* $E \subseteq X \times F$. *A* **factor tree** *is a factor graph $G$ where $G$ is a tree. The* **neighbor set** $\mathcal{N}(v)$ *of a vertex $v$ is the (index-ordered) set of vertices adjacent to vertex $v$. The graph $G$* **represents** *the function $g(X) = \prod_j f_j(X_j)$ if, for each factor $f_j$, the arguments of $f_j$ are its neighbors in $G$, i.e., $\mathcal{N}(f_j) = X_j$.*

Other types of graphical models, such as Bayes' nets [14], can be easily converted into a factor graph representation. When the Bayes' net is a polytree (singly connected directed acyclic graph), the resulting factor graph is a factor tree.

**The Sum-Product Algorithm.** The factorization of $g(X)$ and its structure as represented by the graph $G$ can be used to organize various computations about $g(X)$ efficiently. For example, the marginals of $g(X)$, defined for each $i$ by $g^i(x_i) = \sum_{X \setminus \{x_i\}} g(X)$ can be computed using the sum–product algorithm.

Sum-product is best described in terms of messages sent between each pair of adjacent vertices in the factor graph. For every pair of neighboring vertices $(x_i, f_j) \in E$, the vertex $x_i$ sends a message $\mu_{x_i \to f_j}$ as soon as it receives the messages from all of its neighbors except for $f_j$, and similarly for the message from $f_j$ to $x_i$. The messages between these vertices take the form of a real-valued function of the variable $x_i$; for discrete-valued $x_i$ this can be represented as a vector of length $|A_i|$.

The message $\mu_{x_i \to f_j}$ sent from a variable vertex $x_i$ to a neighboring factor vertex $f_j$, and the message $\mu_{f_j \to x_i}$ from factor $f_j$ to variable $x_i$ are given by

$$\mu_{x_i \to f_j}(x_i) = \prod_{f \in \mathcal{N}(x_i) \setminus f_j} \mu_{f \to x_i}(x_i) \qquad \mu_{f_j \to x_i}(x_i) = \sum_{X_j \setminus x_i} f_j(X_j) \prod_{x \in X_j \setminus x_i} \mu_{x \to f_j}(x)$$

Once all the messages ($2\,|E|$ in total) are sent, we can calculate the marginal $g^i(x_i)$ by simply multiplying all the incoming messages, i.e., $g^i(x_i) = \prod_{f \in \mathcal{N}(x_i)} \mu_{f \to x_i}(x_i)$. Sum–product can be thought of as selecting an efficient elimination ordering of variables (leaf to root) and marginalizing in that order.

**Other Inferences.** Although in this paper we focus on marginal computations using sum–product, similar message passing operations can be generalized to other tasks. For example, the operations of sum–product can be used to compute the data likelihood of any observed evidence; such computations are an inherent part of learning and model comparisons (e.g., [12]). More generally, similar algorithms can be defined to compute functions over any semi–ring possessing the distributive property [11]. Most commonly, the max operation produces a dynamic programming algorithm ("max–product") to compute joint MAP configurations [15].

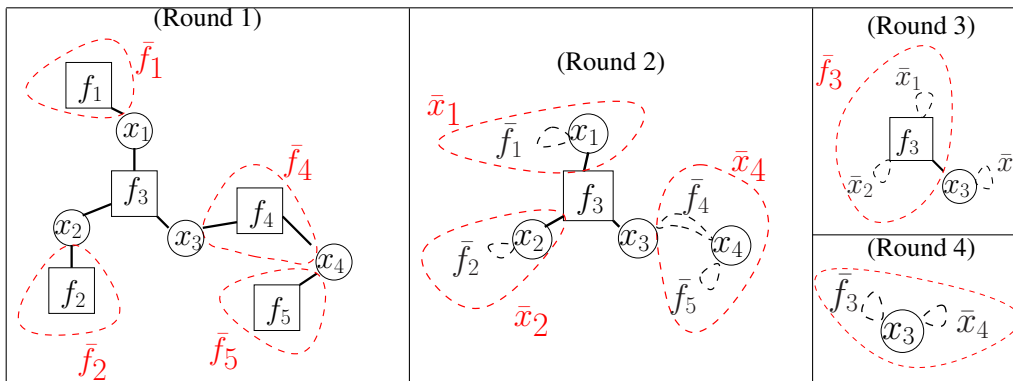

Figure 1: Clustering a factor graph with `rake, compress, finalize` operations.

## 3 Constructing the Cluster Tree

In this section, we describe an algorithm for constructing a balanced representation of the input graphical model, that we call a *cluster tree*. Given the input graphical model, we first apply a clustering algorithm that hierarchically clusters the graphical model, and then apply a labeling algorithm that labels the clusters with *cluster functions* that can be used to compute marginal queries.

**Clustering Algorithm.** Given a factor graph as input, we first tag each node $v$ with a *unary* cluster that consists of $v$ and each edge $(u, v)$ with a *binary* cluster that consists of the edge $(u, v)$. We then cluster the tree hierarchically by applying the *rake*, *compress*, and *finalize* operations. When applied to a leaf node $v$ with neighbor $u$, the *rake* operation deletes the $v$ and the edge $(u, v)$, and forms unary cluster by combining the clusters which tag either $v$ or $(u, v)$; $u$ is tagged with the resulting cluster. When applied to a degree-two node $v$ with neighbors $u$ and $w$, a *compress* operation deletes $v$ and the edges $(u, v)$ and $(v, w)$, inserts the edge $(u, w)$, and forms a binary cluster by combining the clusters which tag the deleted node and edges; $(u, w)$ is then tagged with the resulting cluster. A *finalize* operation is applied when the tree consists of a single node (when no edges remain); it constructs a final cluster that consists of all the clusters with which the final node is tagged.

We cluster a tree $T$ by applying rake and compress operations in rounds. Each round consists of the following two steps until no more edges remain: (1) Apply the rake operation to each leaf; (2) Apply the compress operation to an independent set of degree-two nodes. We choose a random independent set: we flip a coin for each node in each round and apply compress to a degree-two node only if it flips heads and its two neighbors flips tails. This ensures that no two adjacent nodes apply compress simultaneously. When all edges are deleted, we complete the clustering by applying the finalize operation.

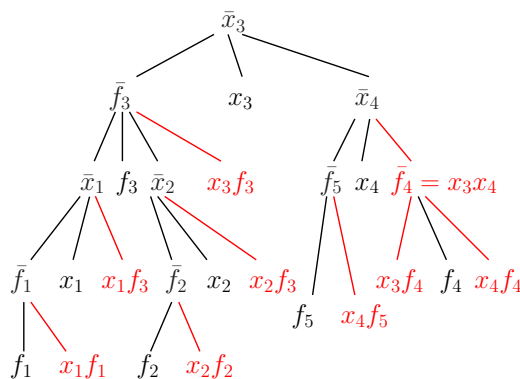

Figure 2: A cluster tree.

Fig. 1 shows a four-round clustering of a factor graph and Fig. 2 shows the corresponding *cluster tree*. In round 1, nodes $f_1, f_2, f_5$ are raked and $f_4$ is compressed. In round 2, $x_1, x_2, x_4$ are raked. In round 3, $f_3$ is raked. A finalize operation is applied in round 4 to produce the final cluster. The leaves of the cluster tree correspond to the nodes and the edges of the factor graph. Each internal node $\bar{v}$ corresponds a unary or a binary cluster formed by deleting $v$. The children of an internal node are the edges and the nodes deleted during the operation that forms the cluster. For example, the cluster $\bar{f}_1$ is formed by the rake operation applied to $f_1$ in round 1. The children of $\bar{f}_1$ are node $f_1$ and edge $(f_1, x_1)$, which are deleted during that operation.

**Labeling Algorithm.** After building the cluster tree, we compute cluster functions along with a notion of orientation for neighboring clusters in a second pass, which we call *labeling*.[1] The cluster function at a node $\bar{v}$ in the tree is computed recursively using the cluster functions at $\bar{v}$'s child clusters, which we denote $S_{\bar{v}} = \{\bar{v}_1, \ldots, \bar{v}_k\}$. Intuitively, each cluster function corresponds to a partial marginalization of the factors contained in cluster $\bar{v}$.

Since each cluster function is defined over a subset of the variables in the original graph, we require some additional notation to represent these sets. Specifically, for a cluster $\bar{v}$, let $A(\bar{v})$ be the arguments of its cluster function, and let $\mathcal{V}(\bar{v})$ be the set of all arguments of its children, $\mathcal{V}(\bar{v}) = \bigcup_i A(\bar{v}_i)$. In a slight abuse of notation, we let $A(v)$ be the arguments of the node $v$ in the original graph, so that if $v$ is a variable node $A(v) = v$ and if $v$ is a factor node $A(v) = \mathcal{N}(v)$.

The cluster functions $c_{\bar{v}}(\cdot)$ and their arguments are then defined recursively, as follows. For the base case, the leaf nodes of the cluster tree correspond to nodes $v$ in the original graph, and we define $c_v$ using the original variables and factors. If $v$ is a factor node $f_j$, we take $c_v(A(v)) = f_j(X_j)$, and if $v$ is a variable node $x_i$, $A(v) = x_i$ and $c_v = 1$. For nodes of the cluster tree corresponding to edges $(u, v)$ of the original graph, we simply take $A(u, v) = \emptyset$ and $c_{u,v} = 1$.

The cluster function at an internal node of the cluster tree is then given by combining the cluster functions of its children and marginalizing over as many variables as possible. Let $\bar{v}$ be the internal node corresponding to the removal of $v$ in the original graph. If $\bar{v}$ is a binary cluster $(u, w)$, that is, at $v$'s removal it had two neighbors $u$ and $w$, then $c_{\bar{v}}$ is given by

$$c_{\bar{v}}(A(\bar{v})) = \sum_{\mathcal{V}(\bar{v}) \setminus A(\bar{v})} \prod_{\bar{v}_i \in S_{\bar{v}}} c_{\bar{v}_i}(A(\bar{v}_i))$$

where the arguments $A(\bar{v}) = \mathcal{V}(\bar{v}) \cap (A(u) \cup A(w))$. For unary cluster $\bar{v}$, where $v$ had a single neighbor $u$ at its removal, $c_{\bar{v}}(\cdot)$ is calculated in the same way with $A(w) = \emptyset$.

We also compute an *orientation* for each cluster's neighbors based on their proximity to the cluster tree's root. This is also calculated recursively using the orientations of each node's ancestors. For a unary cluster $\bar{v}$ with parent $\bar{u}$ in the cluster tree, we define $\text{in}(\bar{v}) = \bar{u}$. For a binary cluster $\bar{v}$ with neighbors $u, w$ at $v$'s removal, define $\text{in}(\bar{v}) = \bar{w}$ and $\text{out}(\bar{v}) = \bar{u}$ if $\bar{w} = \text{in}(\bar{u})$; otherwise $\text{in}(\bar{v}) = \bar{u}$ and $\text{out}(\bar{v}) = \bar{w}$.

We now describe the efficiency of our clustering and labeling algorithms and show that the resulting cluster tree is linear in the size of the input factor graph.

**Theorem 1** (**Hierarchical Clustering**). *A factor tree of $n$ nodes with maximum degree of $k$ can be clustered and labeled in expected $O(d^{k+2}n)$ time where $d$ is the domain size of each variable in the factor tree. The resulting cluster tree has exactly $2n - 1$ leaves and $n$ internal clusters (nodes) and expected $O(\log n)$ depth where the expectation is taken over internal randomization (over the coin flips). Furthermore, the cluster tree has the following properties: (1) each cluster has at most $k + 1$ children, and (2) if $\bar{v} = (u, w)$ is a binary cluster, then $\bar{u}$ and $\bar{w}$ are ancestors of $\bar{v}$, and one of them is the parent of $\bar{v}$.*

*Proof.* Consider first the construction of the cluster tree. The time and the depth bound follow from previous work [2]. The bound on the number of nodes holds because the leaves of the cluster tree correspond to the $n - 1$ edges and $n$ nodes of the factor graph. To see that each cluster has at most $k + 1$ children, note that the a rake or compress operation deletes one node and the at most $k$ edges incident on that node. Every edge appearing in any level of the tree contraction algorithm is represented as a binary cluster $\bar{v} = (u, w)$ in the cluster tree. Whenever an edge is removed, one of the nodes incident to that edge, say $u$ is also removed, making $\bar{u}$ the parent of $\bar{v}$. The fact that $\bar{w}$ is also an ancestor of $\bar{v}$ follows from an induction argument on the levels.

Consider the labeling step. By inspection of the labeling algorithm, the computation of the arguments $A(\cdot)$ and $\mathcal{V}(\cdot)$ requires $O(k)$ time. To bound the time for computing a cluster function, observe that $A(\bar{v})$ is always a singleton set if $\bar{v}$ is a unary cluster, and $A(\bar{v})$ has at most two variables if $\bar{v}$ is a binary cluster. Therefore, $|\mathcal{V}(\bar{v})| \leq k + 2$. The number of operations required to compute

the cluster function at $\bar{v}$ is bounded by $O(|S_{\bar{v}}| d^{|\mathcal{V}(\bar{v})|})$, where $S_{\bar{v}}$ are the children of $\bar{v}$. Since each cluster can appear only once as a child, $\sum |S_{\bar{v}}|$ is $O(n)$ and thus the labeling step takes $O(d^{k+2}n)$ time. Although the running time may appear large, note that the representation of the factor graph takes $O(d^k n)$ space if functions associated with factors are given explicitly. $\qquad\square$

## 4    Queries and Dynamic Changes

We give algorithms for computing marginal queries on the cluster trees and restructuring the cluster tree with respect to changes in the underlying graphical model. For all of these operations, our algorithms require expected logarithmic time in the size of the graphical model.

**Queries.**    We answer marginal queries at a vertex $v$ of the graphical node by using the cluster tree. At a high level, the idea is to find the leaf of the cluster tree corresponding to $v$ and compute the messages along the path from the root of the cluster tree to $v$. Using the orientations computed during the tagging pass, for each cluster $\bar{v}$ we define the following messages:

$$m_{\bar{u}\to\bar{v}} = \begin{cases} \sum_{\mathcal{V}(\bar{u})\setminus A(\bar{v})} \left( m_{\text{in}(\bar{u})\to\bar{u}} \prod_{\bar{u}_i \in S_{\bar{u}}\setminus\{\bar{v}\}} c_{\bar{u}_i}(A(\bar{u}_i)) \right), & \text{if } \bar{u} = \text{in}(\bar{v}) \\[2mm] \sum_{\mathcal{V}(\bar{u})\setminus A(\bar{v})} \left( m_{\text{out}(\bar{u})\to\bar{u}} \prod_{\bar{u}_i \in S_{\bar{u}}\setminus\{\bar{v}\}} c_{\bar{u}_i}(A(\bar{u}_i)) \right), & \text{if } \bar{u} = \text{out}(\bar{v}), \end{cases}$$

where $S_{\bar{u}}$ is the set of the children of $\bar{u}$. Note that for unary clusters, $\text{out}(\cdot)$ is undefined; we define the message in this case to be 1.

**Theorem 2** (**Query**). *Given a factor tree with $n$ nodes, maximum degree $k$, domain size $d$, and its cluster tree, the marginal at any $x_i$ can be computed with the following formula*

$$g^i(x_i) = \sum_{\mathcal{V}(\bar{x}_i)\setminus\{x_i\}} m_{\text{out}(x_i)\to x_i}\, m_{\text{in}(x_i)\to x_i} \prod_{\bar{v}_i \in S_{\bar{x}_i}} c_{\bar{v}_i}(A(\bar{v}_i)),$$

*where $S_{\bar{x}_i}$ is the set of children of $\bar{x}_i$, in $O(kd^{k+2}\log n)$ time.*

Messages are computed only at the ancestors of the query node $x_i$ and downward along the path to $x_i$; there are at most $O(\log n)$ nodes in this path by Theorem 1. Computing each message requires at most $O(kd^{k+2})$ time, and any marginal query takes $O(kd^{k+2}\log n)$ time.

**Dynamic Updates.**    Given a factor graph and its cluster tree, we can change the function of a factor and update the cluster tree by starting at the leaf of the cluster tree that corresponds to the factor and relabeling all the clusters on the path to the root. Updating these labels suffices, because the label of a cluster is a function of its children only. Since relabeling a cluster takes $O(kd^{k+2})$ time and the cluster tree has expected $O(\log n)$ depth, any update requires $O(kd^{k+2}\log n)$ time.

To allow changes to the factor graph itself by insertion/deletion of edges, we maintain a forest of factor trees and the corresponding cluster trees (obtained by clustering the trees one by one). We also maintain the sequence of operations used to construct each cluster tree, i.e., a data structure which represents the state of the clustering at each round. Note that this structure is also size $O(n)$, since at each round a constant fraction of nodes are removed (raked or compressed) in expectation.

Suppose now that the user inserts an edge that connects two trees, or deletes an edge connecting two subtrees. It turns out that both operations have only a limited effect on the sequence of clustering operations performed during construction, affecting only a constant number of nodes at each round of the process. Using a general-purpose *change propagation* technique (detailed in previous work [2, 1]) the necessary alterations can be made to the cluster tree in expected $O(\log n)$ time. Change propagation gives us a new cluster tree that corresponds to the cluster tree that we would have obtained by re-clustering from scratch, conditioned on the same internal randomization process.

In addition to changing the structure of the cluster tree via change propagation, we must also change the labeling information (cluster functions and orientation) of the affected nodes, which can be done using the same process described in Sec. 3. It is a property of the tree contraction process that all such affected clusters form a subtree of the cluster tree that includes the root. Since change propagation affects an expected $O(\log n)$ clusters, and since each cluster can be labeled in $O(kd^{k+2})$ time, the new labels can be computed in $O(kd^{k+2}\log n)$ time.

For dynamic updates, we thus have the following theorem.

**Theorem 3** (**Dynamic Updates**). *For a factor forest $F$ of $n$ vertices with maximum degree $k$, and domain size $d$, the forest of cluster trees can be updated in expected $O(kd^{k+2} \log n)$ time under edge insertions/deletions, and changes to factors.*

## 5    Implementation and Experimental Results

We have implemented our algorithm in Matlab[2] and compared its performance against the standard two-pass sum-product algorithm (used to recompute marginals after dynamic changes). Fig. 3 shows the results of a simulation experiment in which we considered randomly generated factor trees between 100 and 1000 nodes, with each variable having $5^1$, $5^2$, or $5^3$ states, so that each factor has size between $5^2$ and $5^6$. These factor tree correspond roughly to the junction trees of models with between 200 and 6000 nodes, where each node has up to 5 states. Our results show that the time required to build the cluster tree is comparable to one run of sum-product. Furthermore, the query and update operations in the cluster tree incur relatively small constant factors in their asymptotic running time, and are between one to two orders of magnitude faster than recomputing from scratch.

A particularly compelling application area, and one of the original motivations for developing our algorithm, is in the analysis of protein structure. Graphical models constructed from protein structures have recently been used to successfully predict structural properties [17] as well as free energy [9]. These models are typically constructed by taking each node as an amino acid whose states represent their most common conformations, known as *rotamers* [7], and basing conditional probabilities on proximity, and a physical energy function (e.g., [16]) and/or empirical data [4].

Our algorithm is a natural choice for problems where various aspects of protein structure are allowed to change. One such application is *computational mutagenesis*, in which functional amino acids in a protein structure are identified by examining systematic amino acid mutations in the protein structure (i.e., to characterize when a protein "loses" function). In this setting, performing updates to the model (i.e., mutations) and queries (i.e., the free energy or maximum likelihood set of rotameric states) to determine the effect of updates would be likely be far more efficient than standard methods. Thus, our algorithm has the potential to substantially speed up computational studies that examine each of a large number local changes to protein structure, such as in the study of protein flexibility and dynamics. Interestingly, [6] actually give a sample application in computational biology, although their model is a simple sequence-based HMM in which they consider the effect of changing observed sequence on secondary structure only.

The simulation results given in Fig. 3 validate the use of our algorithm for these applications, since protein-structure based graphical models have similar complexity to the inputs we consider: proteins range in size from hundreds to thousands of amino acids, and each amino acid typically has relatively few rotameric states and local interactions. As in prior work [17], our simulation considers a small number of rotamers per amino acid, but the one to two order-of-magnitude speedups obtained by our algorithm indicate that it maybe be possible also to handle higher-resolution models (e.g., with more rotamer states, or degrees of freedom in the protein backbone).

## 6    Conclusion

We give an algorithm for adaptive inference in dynamically changing tree-structured Bayesian networks. Given $n$ nodes in the network, our algorithm can support updates to the observed evidence, conditional probability distributions, as well as updates to the network structure (as long as they keep the network tree-structured) with $O(n)$ preprocessing time and $O(\log n)$ time for queries on any marginal distribution. Our algorithm can easily be modified to maintain MAP estimates as well as compute data likelihoods dynamically, with the same time bounds. We implement the algorithm and show that it can speed up Bayesian inference by orders of magnitude after small updates to the network. Applying our algorithm on the junction tree representation of a graph yields an inference algorithm that can handle updates on conditional distributions and observed evidence in general Bayesian networks (e.g., with cycles); an interesting open question is whether updates to the network structure (i.e., edge insertions/deletions) can also be supported.

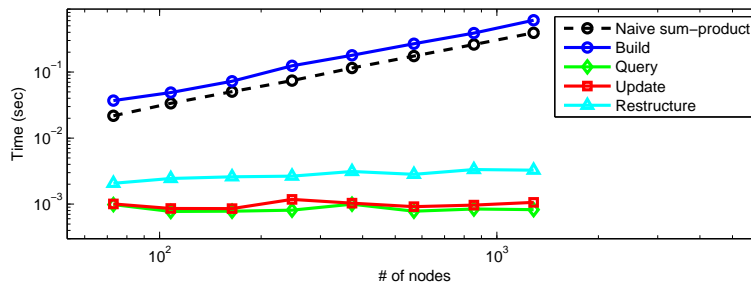

Figure 3: Log-log plot of run time for naive sum-product, building the cluster tree, computing queries, updating factors, and restructuring (adding and deleting edges). Although building the cluster tree is slightly more expensive than sum-product, each subsequent update and query is between 10 and 100 times more efficient than recomputing from scratch.

## Footnotes

[1] Although presented here as a separate labeling operation, the cluster functions can alternatively be computed at the time of the rake or compress operation, as they depend only on the children of $\bar{v}$, and the orientations can be computed during the query operation, since they depend only on the ancestors of $\bar{v}$.

[2]Available for download at *http://www.ics.uci.edu/∼ihler/code/*.

## References

[1] Umut A. Acar, Guy E. Blelloch, Matthias Blume, and Kanat Tangwongsan. An experimental analysis of self-adjusting computation. In *Proceedings of the ACM SIGPLAN Conference on Programming Language Design and Implementation (PLDI)*, 2006.

[2] Umut A. Acar, Guy E. Blelloch, Robert Harper, Jorge L. Vittes, and Maverick Woo. Dynamizing static algorithms with applications to dynamic trees and history independence. In *ACM-SIAM Symposium on Discrete Algorithms (SODA)*, 2004.

[3] Umut A. Acar, Guy E. Blelloch, and Jorge L. Vittes. An experimental analysis of change propagation in dynamic trees. In *Workshop on Algorithm Engineering and Experimentation (ALENEX)*, 2005.

[4] H. M. Berman, J. Westbrook, Z. Feng, G. Gilliland, T. N. Bhat, H. Weissig, I. N. Shindyalov, and P. E. Bourne. The protein data bank. *Nucl. Acids Res.*, 28:235–242, 2000.

[5] P. Clifford. Markov random fields in statistics. In G. R. Grimmett and D. J. A. Welsh, editors, *Disorder in Physical Systems*, pages 19–32. Oxford University Press, Oxford, 1990.

[6] A. L. Delcher, A. J. Grove, S. Kasif, and J. Pearl. Logarithmic-time updates and queries in probabilistic networks. *Journal of Artificial Intelligence Research*, 4:37–59, 1995.

[7] R. L. Dunbrack Jr. Rotamer libraries in the 21st century. *Curr Opin Struct Biol*, 12(4):431–440, 2002.

[8] M. I. Jordan. Graphical models. *Statistical Science*, 19:140–155, 2004.

[9] H. Kamisetty, E. P Xing, and C. J. Langmead. Free energy estimates of all-atom protein structures using generalized belief propagation. In *Proceedings of the 11th Annual International Conference on Research in Computational Molecular Biology*, 2007. To appear.

[10] F. Kschischang, B. Frey, and H.-A. Loeliger. Factor graphs and the sum-product algorithm. *IEEE Transactions on Information Theory*, 47:498–519, 2001.

[11] R. McEliece and S. M. Aji. The generalized distributive law. *IEEE Transactions on Information Theory*, 46(2):325–343, March 2000.

[12] Marina Meilă and Michael I. Jordan. Learning with mixtures of trees. *Journal of Machine Learning Research*, 1(1):1–48, October 2000.

[13] Gary L. Miller and John H. Reif. Parallel tree contraction and its application. In *Proceedings of the 26th Annual IEEE Symposium on Foundations of Computer Science*, pages 487–489, 1985.

[14] J. Pearl. *Probabilistic Reasoning in Intelligent Systems: Networks of Plausible Inference*. Morgan Kaufmann, San Francisco, 1988.

[15] M. J. Wainwright, T. Jaakkola, and A. S. Willsky. Tree consistency and bounds on the performance of the max-product algorithm and its generalizations. *Statistics and Computing*, 14:143–166, April 2004.

[16] S. J. Weiner, P.A. Kollman, D.A. Case, U.C. Singh, G. Alagona, S. Profeta Jr., and P. Weiner. A new force field for the molecular mechanical simulation of nucleic acids and proteins. *J. Am. Chem. Soc.*, 106:765–784, 1984.

[17] C. Yanover and Y. Weiss. Approximate inference and protein folding. In *Proceedings of Neural Information Processing Systems Conference*, pages 84–86, 2002.

